# Incorporating Test Inputs into Learning

**Zehra Cataltepe**
Learning Systems Group
Department of Computer Science
California Institute of Technology
Pasadena, CA 91125
zehra@cs.caltech.edu

**Malik Magdon-Ismail**
Learning Systems Group
Department of Electrical Engineering
California Institute of Technology
Pasadena, CA 91125
magdon@cco.caltech.edu

## Abstract

In many applications, such as credit default prediction and medical image recognition, test inputs are available in addition to the labeled training examples. We propose a method to incorporate the test inputs into learning. Our method results in solutions having smaller test errors than that of simple training solution, especially for noisy problems or small training sets.

## 1 Introduction

We introduce an estimator of test error that takes into consideration the test inputs. The new estimator, augmented error, is composed of the training error and an additional term computed using the test inputs. In some applications, such as credit default prediction and medical image recognition, we do have access to the test inputs. In our experiments, we found that the augmented error (which is computed without looking at the test outputs but only test inputs and training examples) can result in a smaller test error. In particular, it tends to increase when the test error increases (overtraining) even if the simple training error does not. (see figure (1)).

In this paper, we provide an analytic solution for incorporating test inputs into learning in the case of linear, noisy targets and linear hypothesis functions. We also show experimental results for the nonlinear case.

Previous results on the use of unlabeled inputs include Castelli and Cover [2] who show that the labeled examples are exponentially more valuable than unlabeled examples in reducing the classification error. For mixture models, Shahshahani and Landgrebe [7] and Miller and Uyar [6] investigate incorporating unlabeled examples into learning for classification problems and using EM algorithm, and show that unlabeled examples are useful especially when input dimensionality is high and the number of examples is small. In our work we only concentrate on estimating the test error better using the test inputs and our method

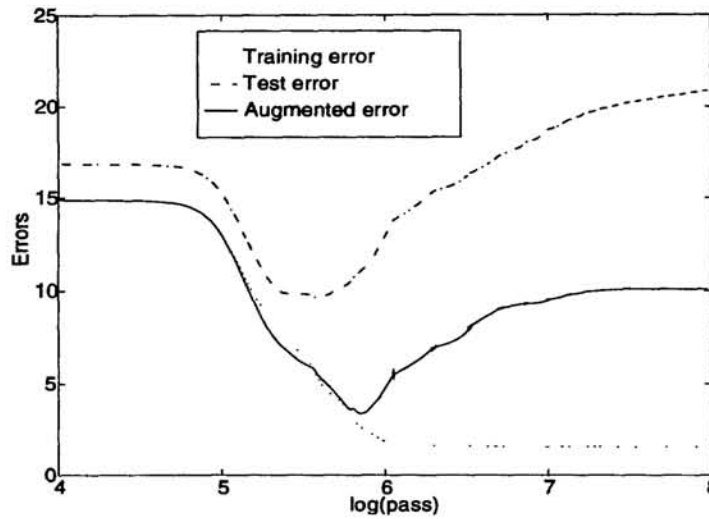

Figure 1: The augmented error, computed not looking at the test outputs at all, follows the test error as overtraining occurs.

extends to the case of unlabeled inputs or input distribution information. Our method is also applicable for regression or classification problems.

In figure 1, we show the training, test and augmented errors, while learning a nonlinear noisy target function with a nonlinear hypothesis. As overtraining occurs, the augmented error follows the test error. In section 2, we explain our method of incorporating test inputs into learning and give the analytical solutions for linear target and hypothesis functions. Section 3 includes theory about the existence and general form of the new solution. Section 4 discusses experimental results. Section 5 extends our solution to the case of knowing the input distribution, or knowing extra inputs that are not necessarily test inputs.

## 2   Incorporating Test Inputs into Learning

In learning-from-examples, we assume we have a **training set**: $\{(\mathbf{x}_1, f_1), \ldots, (\mathbf{x}_N, f_N)\}$ with inputs $\mathbf{x}_n$ and possibly noisy targets $f_n$. Our goal is to choose a hypothesis $g_\mathbf{v}$, among a class of hypotheses $G$, minimizing the test error on an unknown **test set** $\{(\mathbf{y}_1, h_1), \ldots, (\mathbf{y}_M, h_M)\}$.

Using the sample mean square error as our error criterion, the **training error** of hypothesis $g_\mathbf{v}$ is:

$$E_0(g_\mathbf{v}) \;=\; \frac{1}{N} \sum_{n=1}^{N} (g_\mathbf{v}(\mathbf{x}_n) - f_n)^2$$

Similarly the **test error** of $g_\mathbf{v}$ is:

$$E(g_\mathbf{v}) \;=\; \frac{1}{M} \sum_{m=1}^{M} (g_\mathbf{v}(\mathbf{y}_m) - h_m)^2$$

Expanding the test error:

$$E(g_\mathbf{v}) \;=\; \frac{1}{M} \sum_{m=1}^{M} g_\mathbf{v}^2(\mathbf{y}_m) - \frac{2}{M} \sum_{m=1}^{M} g_\mathbf{v}(\mathbf{y}_m) h_m + \frac{1}{M} \sum_{m=1}^{M} h_m^2$$

The main observation is that, when we know the test inputs, we know the first term exactly. Therefore we need only approximate the remaining terms using the training set:

$$E\left(g_{\mathbf{v}}\right) \approx \frac{1}{M}\sum_{m=1}^{M} g_{\mathbf{v}}^2\left(\mathbf{y}_m\right) - \frac{2}{N}\sum_{n=1}^{N} g_{\mathbf{v}}\left(\mathbf{x}_n\right) f_n + \frac{1}{N}\sum_{n=1}^{N} f_n^2 \qquad (1)$$

$$= E_0\left(g_{\mathbf{v}}\right) + \frac{1}{M}\sum_{m=1}^{M} g_{\mathbf{v}}^2\left(\mathbf{y}_m\right) - \frac{1}{N}\sum_{n=1}^{N} g_{\mathbf{v}}^2\left(\mathbf{x}_n\right)$$

We scale the addition to the training error by an **augmentation parameter** $\alpha$ to obtain a more general error function that we call the **augmented error**:

$$E_\alpha\left(g_{\mathbf{v}}\right) = E_0\left(g_{\mathbf{v}}\right) + \alpha\left(\frac{1}{M}\sum_{m=1}^{M} g_{\mathbf{v}}^2\left(\mathbf{y}_m\right) - \frac{1}{N}\sum_{n=1}^{N} g_{\mathbf{v}}^2\left(\mathbf{x}_n\right)\right)$$

where $\alpha = 0$ corresponds to the training error $E_0$ and $\alpha = 1$ corresponds to equation (1).

The best value of the augmentation parameter depends on a number of factors including the target function, the noise distribution and the hypothesis class. In the following sections we investigate properties of the best augmentation parameter and give a method of finding the best augmentation parameter when the hypothesis is linear.

## 3  Augmented Solution for the Linear Hypothesis

In this section we assume hypothesis functions of the form $g_{\mathbf{v}}(\mathbf{x}) = \mathbf{v}^T\mathbf{x}$. From here onwards we will denote the functions by the vector that multiplies the inputs. When the hypothesis is linear we can find the minimum of the augmented error analytically.

Let $X_{d\times N}$ be the matrix of training inputs, $Y_{d\times M}$ be the matrix of test inputs and $\mathbf{f}_{N\times 1}$ contain the training targets. The solution $\mathbf{w}_0$ minimizing the training error $E_0$ is the least squares solution [5]: $\mathbf{w}_0 = \left(\frac{XX^T}{N}\right)^{-1}\frac{X\mathbf{f}}{N}$.

The augmented error $E_\alpha\left(\mathbf{v}\right) = E_0\left(\mathbf{v}\right) + \alpha\mathbf{v}^T\left(\frac{YY^T}{M} - \frac{XX^T}{N}\right)\mathbf{v}$ is minimized at the augmented error $\mathbf{w}_\alpha$:

$$\mathbf{w}_\alpha = \left(I - \alpha R\right)^{-1}\mathbf{w}_0 \qquad (2)$$

where $R = I - \left(\frac{XX^T}{N}\right)^{-1}\frac{YY^T}{M}$. When $\alpha = 0$, the augmented solution $\mathbf{w}_\alpha$ is equal to the least mean squares solution $\mathbf{w}_0$.

## 4  Properties of the Augmentation Parameter

Assume a linear target and possibly noisy training outputs: $\mathbf{f} = \mathbf{w}^{*T}X + \mathbf{e}$ where $\langle\mathbf{e}\mathbf{e}^T\rangle = \sigma_e^2 I_{N\times N}$.

Since the specific realization of noise $\mathbf{e}$ is unknown, instead of minimizing the test error directly, we focus on minimizing $\langle E\left(\mathbf{w}_\alpha\right)\rangle_{\mathbf{e}}$, the expected value of the test error of the augmented solution with respect to the noise distribution:

$$\langle E\left(\mathbf{w}_\alpha\right)\rangle_{\mathbf{e}} = \mathbf{w}^{*T}\left(\left(I - \alpha R^T\right)^{-1} - I\right)\frac{YY^T}{M}\left(\left(I - \alpha R\right)^{-1} - I\right)\mathbf{w}^*$$

$$+ \frac{\sigma_e^2}{N}tr\left(\left(I - \alpha R^T\right)^{-1}\frac{YY^T}{M}\left(I - \alpha R\right)^{-1}\left(\frac{XX^T}{N}\right)^{-1}\right) \qquad (3)$$

where we have used $\langle e^T A e \rangle_e = \sigma_e^2 tr(A)$ and $tr(A)$ denotes the trace of matrix $A$. When $\alpha = 0$, we have:

$$\langle E(\mathbf{w}_0) \rangle_e = \frac{\sigma_e^2}{N} tr\left( \frac{YY^T}{M} \left( \frac{XX^T}{N} \right)^{-1} \right) \tag{4}$$

Now, we prove the existence of a nonzero augmentation parameter $\alpha$ when the outputs are noisy.

**Theorem 1:** If $\sigma_e^2 > 0$ and $tr(R(I - R)) \neq 0$, then there is an $\alpha \neq 0$ that minimizes the expected test error $\langle E(\mathbf{w}_\alpha) \rangle_e$.

**Proof:** Since $\frac{\partial B^{-1}(\alpha)}{\partial \alpha} = -B^{-1}(\alpha) \frac{\partial B(\alpha)}{\partial \alpha} B^{-1}(\alpha)$ for any matrix B whose elements are scalar functions of $\alpha$ [3], the derivative of $\langle E(\mathbf{w}_\alpha) \rangle_e$ with respect to $\alpha$ at $\alpha = 0$ is:

$$\left. \frac{d\langle E(\mathbf{w}_\alpha) \rangle_e}{d\alpha} \right|_{\alpha=0} = 2\frac{\sigma_e^2}{N} tr\left( R\left( \frac{XX^T}{N} \right)^{-1} \frac{YY^T}{M} \right) = 2\frac{\sigma_e^2}{N} tr(R(I - R))$$

If the derivative is $< 0$ ($> 0$ respectively), then $\langle E(\mathbf{w}_\alpha) \rangle_e$ is minimized at some $\alpha > 0$ ($\alpha < 0$ respectively). $\square$

The following proposition gives an approximate formula for the best $\alpha$.

**Theorem 2:** If $N$ and $M$ are large, and the training and test inputs are drawn i.i.d from an input distribution with covariance matrix $\langle \mathbf{x}\mathbf{x}^T \rangle = \sigma_x^2 I$, then the $\alpha^*$ minimizing $\langle E(\mathbf{w}_\alpha) \rangle_{e,\mathbf{x},\mathbf{y}}$, the expected test error of the augmented solution with respect to noise and inputs, is approximately:

$$\alpha^* \approx \frac{d}{N} \frac{\sigma_e^2}{\sigma_x^2 \mathbf{w}^{*T} \mathbf{w}^*} \tag{5}$$

**Proof:** is given in the appendix. $\square$

This formula determines the behavior of the best $\alpha$. The best $\alpha$:

- decreases as the signal-to-noise ratio increases.
- increases as $\frac{d}{N}$ increases, i.e. as we have less examples per input dimension.

### 4.1 $\mathbf{w}_\alpha$ as an Estimator of $\mathbf{w}^*$

The mean squared error (m.s.e.) of any estimator $\hat{\mathbf{w}}$ of $\mathbf{w}^*$, can be written as [1]:

$$\langle \|\mathbf{w}^* - \hat{\mathbf{w}}\|^2 \rangle_e = \|\mathbf{w}^* - \langle\hat{\mathbf{w}}\rangle_e\|^2 + \langle \|\hat{\mathbf{w}} - \langle\hat{\mathbf{w}}\rangle_e\|^2 \rangle_e$$

$$m.s.e(\hat{\mathbf{w}}) = bias^2(\hat{\mathbf{w}}) + variance(\hat{\mathbf{w}})$$

When $\alpha$ is independent of the specific realization e of the noise:

$$m.s.e.(\mathbf{w}_\alpha) = \mathbf{w}^{*T} \left( I - (I - \alpha R^T)^{-1} \right) \left( I - (I - \alpha R)^{-1} \right) \mathbf{w}^*$$

$$+ \frac{\sigma_e^2}{N} tr\left( \left( \frac{XX^T}{N} \right)^{-1} (I - \alpha R^T)^{-1} (I - \alpha R)^{-1} \right)$$

Hence the $m.s.e.$ of the least square estimator $\mathbf{w}_0$ is:

$$m.s.e.(\mathbf{w}_0) = \frac{\sigma_e^2}{N} tr\left(\left(\frac{XX^T}{N}\right)^{-1}\right)$$

$\mathbf{w}_0$ is the minimum variance unbiased linear estimator of $\mathbf{w}^*$. Although $\mathbf{w}_\alpha$ is a biased estimator if $\alpha R \neq 0$, the following proposition shows that, when there is noise, there is an $\alpha \neq 0$ minimizing the $m.s.e.$ of $\mathbf{w}_\alpha$:

**Theorem 3:** If $\sigma_e^2 > 0$ and $tr\left(\left(\frac{XX^T}{N}\right)^{-1}(R + R^T)\right) \neq 0$, then there is an $\alpha \neq 0$ that minimizes the m.s.e. of $\mathbf{w}_\alpha$.

**Proof:** is similar to the proof of proposition 1 and will be skipped $\square$.

As $N$ and $M$ get large, $R = I - \left(\frac{XX^T}{N}\right)^{-1}\frac{YY^T}{M} \to 0$ and $\mathbf{w}_\alpha = (I - \alpha R)^{-1}\mathbf{w}_0 \to \mathbf{w}_0$. Hence, for large $N$ and $M$, the bias and variance of $\mathbf{w}_\alpha$ approach 0, making $\mathbf{w}_\alpha$ an unbiased and consistent estimator of $\mathbf{w}^*$.

# 5    A Method to Find the Best Augmentation Parameter

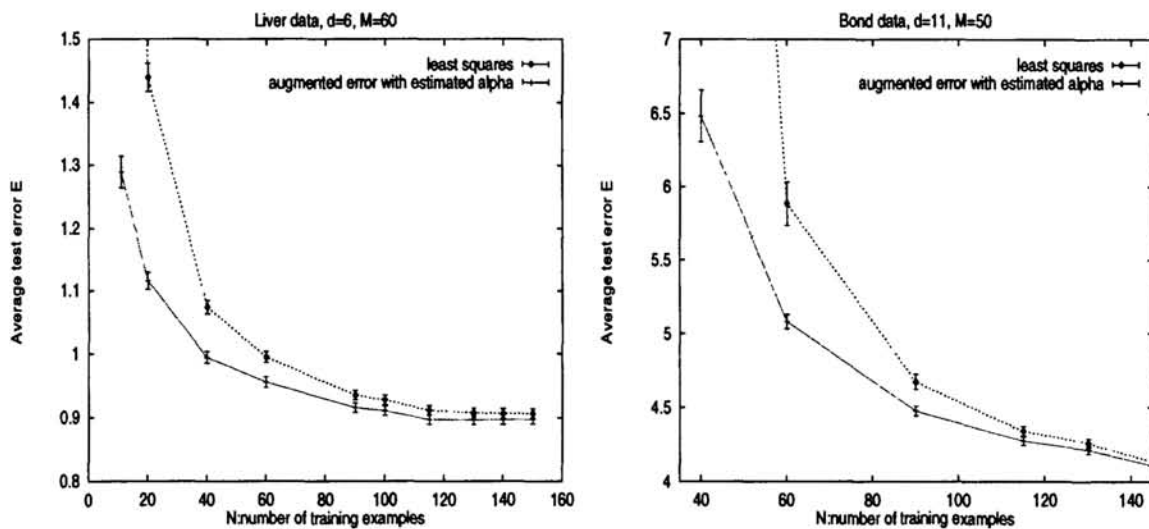

Figure 2: Using the augmented error results in smaller test error especially when the number of training examples is small.

Given only the training and test inputs $X$ and $Y$, and the training outputs $\mathbf{f}$, in this section we propose a method to find the best $\alpha$ minimizing the test error of $\mathbf{w}_\alpha$.

Equation (3) gives a formula for the expected test error which we want to minimize. However, we do not know the target $\mathbf{w}^*$ and the noise variance $\sigma_e^2$. In equation (3), we replace $\mathbf{w}^*$ by $\mathbf{w}_\alpha$ and $\sigma_e^2$ by $\frac{(X^T\mathbf{w}_\alpha - \mathbf{f})^T(X^T\mathbf{w}_\alpha - \mathbf{f})}{N-d-1}$, where $\mathbf{w}_\alpha$ is given by equation (2). Then we find the $\alpha$ minimizing the resulting approximation to the expected test error.

We experimented with this method of finding the best $\alpha$ on artificial and real data. The results of experiments for liver data[1] and bond data[2] are shown in figure 2. In the liver

database the inputs are different blood test results and the output is the number of drinks per day. The bond data consists of financial ratios as inputs and rating of the bond from AAA to B- or lower as the output.

We also compared our method to the least squares ($\mathbf{w}_0$) and early stopping using different validation set sizes for linear and noisy problems. The table below shows the results.

| SNR | mean $\frac{E(\mathbf{w}_\alpha)}{E(\mathbf{w}_0)}$ | mean $\frac{E(\mathbf{w}_{early\ stop})}{E(\mathbf{w}_0)}, N_v = \frac{N}{3}$ | mean $\frac{E(\mathbf{w}_{early\ stop})}{E(\mathbf{w}_0)}, N_v = \frac{N}{6}$ |
|---|---|---|---|
| 0.01 | $0.650 \pm 0.006$ | $0.126 \pm 0.003$ | $0.192 \pm 0.004$ |
| 1 | $0.830 \pm 0.007$ | $1.113 \pm 0.021$ | $1.075 \pm 0.020$ |
| 100 | $1.001 \pm 0.002$ | $2.373 \pm 0.040$ | $2.073 \pm 0.042$ |

Table 1: Augmented solution is consistently better than the least squares whereas early stopping gives worse results as the signal-to-noise ratio (SNR) increases. Even averaging early stopping solutions did not help when SNR = 100 ($\frac{E(\mathbf{w}_{early\ stop})}{E(\mathbf{w}_0)} = 1.245 \pm 0.018$ when $N_v = \frac{N}{3}$ and $1.307 \pm 0.021$ for $N_v = \frac{N}{6}$). For the results shown, $d = 11$, $N = 30$ training examples were used, $N_v$ is the number of validation examples.

## 6  Extensions

When the input probability distribution or the covariance matrix of inputs, instead of test inputs are known, $\frac{YY^T}{M}$ can be replaced by $\langle \mathbf{x}\mathbf{x}^T \rangle = \Sigma$ and our methods are still applicable.

If the inputs available are not test inputs but just some extra inputs, they can still be incorporated into learning. Let us denote the extra $K$ inputs $\{\mathbf{z}_1, \ldots, \mathbf{z}_K\}$ by the matrix $Z_{d \times K}$. Then the augmented error becomes:

$$E_\alpha(\mathbf{v}) = E_0(\mathbf{v}) + \alpha \frac{K}{K+N} \mathbf{v}^T \left( \frac{ZZ^T}{K} - \frac{XX^T}{N} \right) \mathbf{v}$$

The augmented new solution and its expected test error are same as in equations (2) and (3), except we have $R_Z = I - \left( \frac{XX^T}{N} \right)^{-1} \frac{ZZ^T}{K}$ instead of $R$.

Note that for the linear hypothesis case, the augmented error is not necessarily a regularized version of the training error, because the matrix $\frac{YY^T}{M} - \frac{XX^T}{N}$ is not necessarily a positive definite matrix.

## 7  Conclusions and Future Work

We have demonstrated a method of incorporating inputs into learning when the target and hypothesis functions are linear, and the target is noisy. We are currently working on extending our method to nonlinear target and hypothesis functions.

### Appendix

**Proof of Theorem 2:** When the spectral radius of $\alpha R$ is less than 1 ($\alpha$ is small and/or $N$ and $M$ are large), we can approximate $(I - \alpha R)^{-1} \approx I + \alpha R$ [4], and similarly, $(I - \alpha R^T)^{-1} \approx I + \alpha R^T$. Discarding any terms with powers of $\alpha$ greater than 1, and

solving for $\alpha$ in $\frac{d\langle E(\mathbf{w}_\alpha)\rangle_{\bullet,\mathbf{x},\mathbf{y}}}{d\alpha} = \left\langle \frac{d\langle E(\mathbf{w}_\alpha)\rangle_\bullet}{d\alpha} \right\rangle_{\mathbf{x},\mathbf{y}} = 0$:

$$\alpha^* \approx \frac{\sigma_e^2}{N} \frac{\langle tr\left(R(R-I)\right)\rangle_{\mathbf{x},\mathbf{y}}}{\left\langle \mathbf{w}^T \frac{YY^T}{M} R^2 \mathbf{w} \right\rangle_{\mathbf{x},\mathbf{y}}} \approx \frac{\sigma_e^2}{N} \frac{tr\left(\langle R^2 - R\rangle_{\mathbf{x},\mathbf{y}}\right)}{\sigma_x^2 \mathbf{w}^T \langle R^2 \rangle_{\mathbf{x},\mathbf{y}} \mathbf{w}}$$

The last step follows since we can write $\frac{YY^T}{M} = \sigma_x^2 \left(I + \frac{V_y}{\sqrt{M}}\right)$, $\frac{XX^T}{N} = \sigma_x^2 \left(I - \frac{V_x}{\sqrt{N}}\right)$ and $\left(\frac{XX^T}{N}\right)^{-1} = \frac{1}{\sigma_x^2}\left(I + \frac{V_x}{\sqrt{N}} + \frac{V_x^2}{N}\right) + O\left(\frac{1}{N^{1.5}}\right)$ for matrices $V_x$ and $V_y$ such that $\langle V_x \rangle_{\mathbf{x}} = \langle V_y \rangle_{\mathbf{y}} = 0$ and $\langle V_x^2 \rangle_x$ and $\langle V_y^2 \rangle_y$ are constant with respect to $N$ and $M$. For large $M$ we can approximate $\left\langle \frac{YY^T}{M} R^2 \right\rangle_{\mathbf{x},\mathbf{y}} = \sigma_x^2 \langle R^2 \rangle_{\mathbf{x},\mathbf{y}}$.

Ignoring terms of $O\left(\frac{1}{N^{1.5}}\right)$, $\langle R^2 - R \rangle_{\mathbf{x},\mathbf{y}} = \left\langle 2\frac{V_x^2}{N} + \frac{V_y^2}{M} \right\rangle_{\mathbf{x},\mathbf{y}}$ and $\langle R^2 \rangle_{\mathbf{x},\mathbf{y}} = \left\langle \frac{V_x^2}{N} + \frac{V_y^2}{M} \right\rangle_{\mathbf{x},\mathbf{y}}$. It can be shown that $\left\langle \frac{V_x^2}{N} \right\rangle_{\mathbf{x}} = \frac{\lambda}{N}I$ for a constant $\lambda$ depending on the input distribution. Similarly $\left\langle \frac{V_y^2}{M} \right\rangle_{\mathbf{y}} = \frac{\lambda}{M}I$. Therefore:

$$\alpha^* \approx \frac{d}{N} \frac{\sigma_e^2}{\sigma_x^2 \mathbf{w}^T \mathbf{w}} \frac{\frac{2}{N} + \frac{1}{M}}{\frac{1}{N} + \frac{1}{M}} \approx \frac{d}{N} \frac{\sigma_e^2}{\sigma_x^2 \mathbf{w}^T \mathbf{w}}$$

□

## Acknowledgments

We would like to thank the Caltech Learning Systems Group: Prof. Yaser Abu-Mostafa, Dr. Amir Atiya, Alexander Nicholson, Joseph Sill and Xubo Song for many useful discussions.

## Footnotes

[1] ftp://ftp.ics.uci.edu/pub/machine-learning-databases/liver-disorders/bupa.data

[2] We thank Dr. John Moody for providing the bond data.

## References

[1] Bishop, C. (1995) *Neural Networks for Pattern Recognition*, Clarendon Press, Oxford, 1995.

[2] Castelli, V. & Cover T. (1995) On the Exponential Value of Labeled Samples. *Pattern Recognition Letters*, Vol. 16, Jan. 1995, pp. 105-111.

[3] Devijver, P. A. & Kittler, J. (1982) *Pattern Recognition: A Statistical Approach*, pp. 434. Prentice-Hall International, London.

[4] Golub, G. H. & Van Loan C. F. (1993) *Matrix Computations*, The Johns-Hopkins University Press, Baltimore, MD.

[5] Hocking, R. R. (1996) *Methods and Applications of Linear Models*. John Wiley & Sons, NY.

[6] Miller, D. J. & Uyar, S. (1996), A Mixture of Experts Classifier with Learning Based on Both Labeled and Unlabeled Data. In G. Tesauro, D. S. Touretzky and T.K. Leen (eds.), *Advances in Neural Information Processing Systems 9*. Cambridge, MA: MIT Press.

[7] Shahshahani, B. M. & Landgrebe, D. A. (1994) The Effect of Unlabeled Samples in Reducing Small Sample Size Problem and Mitigating the Hughes Phonemenon. *IEEE Transactions on Geoscience and Remote Sensing*, Vol. 32 No. 5, Sept 1994, pp. 1087–1095.